# Context Effects in Category Learning:
# An Investigation of Four Probabilistic Models

**Michael C. Mozer**[+◇]**, Michael Jones**[◇†]**, Michael Shettel**[+]
[+]Dept. of Computer Science, [†]Dept. of Psychology, and [◇]Institute of Cognitive Science
University of Colorado, Boulder, CO 80309-0430
{mozer,mike.jones,shettel}@colorado.edu

## Abstract

Categorization is a central activity of human cognition. When an individual is asked to categorize a sequence of items, context effects arise: categorization of one item influences category decisions for subsequent items. Specifically, when experimental subjects are shown an exemplar of some target category, the category prototype appears to be pulled toward the exemplar, and the prototypes of all nontarget categories appear to be pushed away. These *push* and *pull* effects diminish with experience, and likely reflect long-term learning of category boundaries. We propose and evaluate four principled probabilistic (Bayesian) accounts of context effects in categorization. In all four accounts, the probability of an exemplar given a category is encoded as a Gaussian density in feature space, and categorization involves computing category posteriors given an exemplar. The models differ in how the uncertainty distribution of category prototypes is represented (localist or distributed), and how it is updated following each experience (using a maximum likelihood gradient ascent, or a Kalman filter update). We find that the distributed maximum-likelihood model can explain the key experimental phenomena. Further, the model predicts other phenomena that were confirmed via reanalysis of the experimental data.

Categorization is a key cognitive activity. We continually make decisions about characteristics of objects and individuals: Is the fruit ripe? Does your friend seem unhappy? Is your car tire flat? When an individual is asked to categorize a sequence of items, context effects arise: categorization of one item influences category decisions for subsequent items. Intuitive naturalistic scenarios in which context effects occur are easy to imagine. For example, if one lifts a medium-weight object after lifting a light-weight or heavy-weight object, the medium weight feels heavier following the light weight than following the heavy weight. Although the object-contrast effect might be due to fatigue of sensory-motor systems, many context effects in categorization are purely cognitive and cannot easily be attributed to neural habituation. For example, if you are reviewing a set of conference papers, and the first three in the set are dreadful, then even a mediocre paper seems like it might be above threshold for acceptance. Another example of a category boundary shift due to context is the following. Suppose you move from San Diego to Pittsburgh and notice that your neighbors repeatedly describe muggy, somewhat overcast days as "lovely." Eventually, your notion of what constitutes a lovely day accommodates to your new surroundings.

As we describe shortly, experimental studies have shown a fundamental link between context effects in categorization and long-term learning of category boundaries. We believe that context effects can be viewed as a reflection of a trial-to-trial learning, and the cumulative effect of these trial-to-trial modulations corresponds to what we classically consider to be category learning. Consequently, any compelling model of category learning should also be capable of explaining context effects.

## 1 Experimental Studies of Context Effects in Categorization

Consider a set of stimuli that vary along a single continuous dimension. Throughout this paper, we use as an illustration circles of varying diameters, and assume four categories of circles defined ranges of diameters; call them A, B, C, and D, in order from smallest to largest diameter.

In a *classification* paradigm, experimental subjects are given an exemplar drawn from one category and are asked to respond with the correct category label (Zotov, Jones, & Mewhort, 2003). After making their response, subjects receive feedback as to the correct label, which we'll refer to as the *target*. In a *production* paradigm, subjects are given a target category label and asked to produce an exemplar of that category, e.g., using a computer mouse to indicate the circle diameter (Jones & Mewhort, 2003). Once a response is made, subjects receive feedback as to the correct or *true* category label for the exemplar they produced. Neither classification nor production task has sequential structure, because the order of trial is random in both experiments.

The production task provides direct information about the subjects' internal representations, because subjects are producing exemplars that they consider to be prototypes of a category, whereas the categorization task requires indirect inferences to be made about internal representations from reaction time and accuracy data. Nonetheless, the findings in the production and classification tasks mirror one another nicely, providing converging evidence as to the nature of learning. The production task reveals how mental representations shift as a function of trial-to-trial sequences, and these shifts cause the sequential pattern of errors and response times typically observed in the classification task. We focus on the production task in this paper because it provides a richer source of data. However, we address the categorization task with our models as well.

Figure 1 provides a schematic depiction of the key sequential effects in categorization. The horizontal line represents the stimulus dimension, e.g., circle diameter. The dimension is cut into four regions labeled with the corresponding category. The category center, which we'll refer to as the *prototype*, is indicated by a vertical dashed line. The long solid vertical line marks the current exemplar—whether it is an exemplar presented to subjects in the classification task or an exemplar generated by subjects in the production task. Following an experimental trial with this exemplar, category prototypes appear to shift: the target-category prototype moves toward the exemplar, which we refer to as a *pull* effect, and all nontarget-category prototypes move away from the exemplar, which we refer to as a *push* effect. Push and pull effects are assessed in the production task by examining the exemplar produced on the following trial, and in the categorization task by examining the likelihood of an error response near category boundaries.

The set of phenomena to be explained are as follows, described in terms of the production task. All numerical results referred to are from Jones and Mewhort (2003). This experiment consisted of 12 blocks of 40 trials, with each category label given as target 10 times within a block.

- Within-category pull: When a target category is repeated on successive trials, the exemplar generated on the second trial moves toward the exemplar generated on the first trial, with respect to the true category prototype. Across the experiment, a correlation coefficient of 0.524 is obtained, and remains fairly constant over trials.

- Between-category push: When the target category changes from one trial to the next, the exemplar generated on the second trial moves away from the exemplar generated on the first trial (or equivalently, from the prototype of the target category on the first trial). Figure 2a summarizes the sequential push effects from Jones and Mewhort. The diameter of the circle produced on trial $t$ is plotted as a function of the target category on trial $t-1$, with one line for each of the four trial $t$ targets. The mean diameter for each target category is subtracted out, so the absolute vertical offset of each line is unimportant. The main feature of the data to note is that all four curves have a negative slope, which has the following meaning: the smaller that target $t-1$ is (i.e., the further to the left on the x axis in Figure 1), the larger the response to target $t$ is (further to the right in Figure 1), and vice versa, reflecting a push away from target $t-1$. Interestingly and importantly, the magnitude of the push increases with the ordinal distance between targets $t-1$ and $t$. Figure 2a is based on data from only eight subjects and is therefore noisy, though the effect is statistically reliable. As further evidence, Figure 2b shows data from a categorization task (Zotov et al., 2003), where the y-axis is a different dependent measure, but the negative slope has the same interpretation as in Figure 2a.

Figure 1: Schematic depiction of sequential effects in categorization

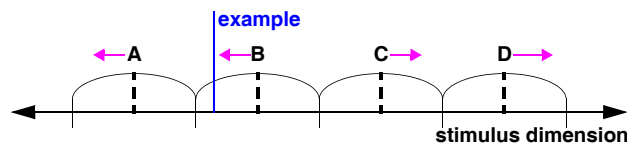

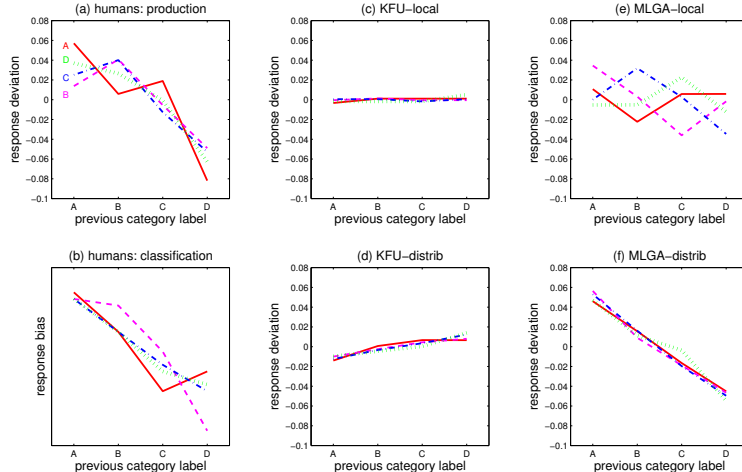

Figure 2: Push effect data from (a) production task of Jones and Mewhort (2003), (b) classification task of Zotov et al. (2003), and (c)-(f) the models proposed in this paper. The y axis is the deviation of the response from the mean, as a proportion of the total category width. The response to category A is solid red, B is dashed magenta, C is dash-dotted blue, and D is dotted green.

- Push and pull effects are not solely a consequence of errors or experimenter feedback. In quantitative estimation of push and pull effects, trial $t$ is included in the data only if the response on trial $t - 1$ is correct. Thus, the effects follow trials in which no error feedback is given to the subjects, and therefore the adjustments are not due to explicit error correction.

- Push and pull effects diminish over the course of the experiment. The magnitude of push effects can be measured by the slope of the regression lines fit to the data in Figure 2a. The slopes get shallower over successive trial blocks. The magnitude of pull effects can be measured by the standard deviation (SD) of the produced exemplars, which also decreases over successive trial blocks.

- Accuracy increases steadily over the course of the experiment, from 78% correct responses in the first block to 91% in the final block. This improvement occurs despite the fact that error feedback is relatively infrequent and becomes even less frequent as performance improves.

## 2   Four Models

In this paper, we explore four probabilistic (Bayesian) models to explain data described in the previous section. The key phenomenon to explain turns out to be the push effect, for which three of the four models fail to account. Modelers typically discard the models that they reject, and present only their pet model. In this work, we find it useful to report on the rejected models for three reasons. First, they help to set up and motivate the one successful model. Second, they include several obvious candidates, and we therefore have the imperative to address them. Third, in order to evaluate a model that can explain certain data, one needs to know the degree to which the the data constrain the space of models. If many models exist that are consistent with the data, one has little reason to prefer our pet candidate.

Underlying all of the models is a generative probabilistic framework in which a category $i$ is represented by a prototype value, $d_i$, on the dimension that discriminates among the categories. In the example used throughout this paper, the dimension is the diameter of a circle (hence the notation $d$ for the prototype). An exemplar, $E$, of category $i$ is drawn from a Gaussian distribution with mean $d_i$ and variance $v_i$, denoted $E \sim \mathcal{N}(d_i, v_i)$. Category learning involves determining $\mathbf{d} \equiv \{d_i\}$. In this work, we assume that the $\{v_i\}$ are fixed and given. Because $\mathbf{d}$ is unknown at the start of the experiment, it is treated as the value of a random vector, $\mathbf{D} \equiv \{D_i\}$. Figure 3a shows a simple graphical model representing the generative framework, in which $E$ is the exemplar and $C$ the category label.

To formalize our discussion so far, we adopt the following notation:

$$P(E|C = c, \mathbf{D} = \mathbf{d}) \sim \mathcal{N}(\mathbf{h}_c\mathbf{d}, v_c), \qquad (1)$$

where, for the time being, $\mathbf{h}_c$ is a unary column vector all of whose elements are zero except for element $c$ which has value 1. (Subscripts may indicate either an index over elements of a vector, or an index over vectors. Boldface is used for vectors and matrices.)

Figure 3: (a) Graphical model depicting selection of an exemplar, $E$, of a category, $C$, based on the prototype vector, $\mathbf{D}$; (b) Dynamic version of model indexed by trials, $t$

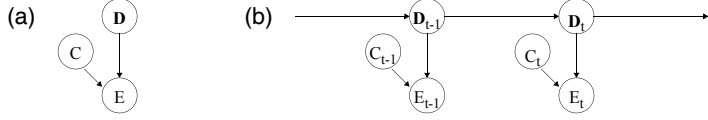

We assume that the prototype representation, $\mathbf{D}$, is multivariate Gaussian, $\mathbf{D} \sim \mathcal{N}(\Psi, \Sigma)$, where $\Psi$ and $\Sigma$ encode knowledge—and uncertainty in the knowledge—of the category prototype structure. Given this formulation, the uncertainty in $\mathbf{D}$ can be integrated out:

$$P(E|C) \sim \mathcal{N}(\mathbf{h}_c\Psi, \mathbf{h}_c\Sigma\mathbf{h}_c^{\mathrm{T}} + v_c). \tag{2}$$

For the categorization task, a category label can be assigned by evaluating the *category posterior*, $P(C|E)$, via Bayes rule, Equation 1, and the category priors, $P(C)$.

In this framework, learning takes place via trial-to-trial adaptation of the category prototype distribution, $\mathbf{D}$. In Figure 3b, we add the subscript $t$ to each random variable to denote the trial, yielding a dynamic graphical model for the sequential updating of the prototype vector, $\mathbf{D}_t$. (The reader should be attentive to the fact that we use subscripted indices to denote both trials and category labels. We generally use the index $t$ to denote trial, and $c$ or $i$ to denote a category label.) The goal of our modeling work is to show that the sequential updating process leads to context effects, such as the push and pull effects discussed earlier.

We propose four alternative models to explore within this framework. The four models are obtained via the Cartesian product of two binary choices: the learning rule and the prototype representation.

## 2.1 Learning rule

The first learning rule, *maximum likelihood gradient ascent* (*MLGA*), attempts to adjust the prototype representation so as to maximize the log posterior of the category given the exemplar. (The category, $C = c$, is the *true* label associated with the exemplar, i.e., either the target label the subject was asked to produce, or—if an error was made—the actual category label the subject did produce.) Gradient ascent is performed in all parameters of $\Psi$ and $\Sigma$:

$$\Delta\psi_i = \epsilon_\psi \frac{\partial}{\partial\psi_i} log(P(c|e)) \quad \text{and} \quad \Delta\sigma_{ij} = \epsilon_\sigma \frac{\partial}{\partial\sigma_{ij}} log(P(c|e)), \tag{3}$$

where $\epsilon_\psi$ and $\epsilon_\sigma$ are step sizes. To ensure that $\Sigma$ remains a covariance matrix, constrained gradient steps are applied. The constraints are: (1) diagonal terms are nonnegative, i.e., $\sigma_i^2 \geq 0$; (2) off-diagonal terms are symmetric, i.e., $\sigma_{ij} = \sigma_{ji}$; and (3) the matrix remains positive definite, ensured by $-1 \leq \frac{\sigma_{ij}}{\sigma_i\sigma_j} \leq 1$.

The second learning rule, a *Kalman filter update* (*KFU*), reestimates the uncertainty distribution of the prototypes given evidence provided by the current exemplar and category label. To draw the correspondence between our framework and a Kalman filter: the exemplar is a scalar measurement that pops out of the filter, the category prototypes are the hidden state of the filter, the measurement noise is $v_c$, and the linear mapping from state to measurement is achieved by $\mathbf{h}_c$. Technically, the model is a *measurement-switched* Kalman filter, where the switching is determined by the category label $c$, i.e., the measurement function, $\mathbf{h}_c$, and noise, $v_c$, are conditioned on $c$. The Kalman filter also allows temporal dynamics via the update equation, $\mathbf{d}_t = A\mathbf{d}_{t-1}$, as well as internal process noise, whose covariance matrix is often denoted $Q$ in standard Kalman filter notation. We investigated the choice of $A$ and $R$, but because they did not impact the qualitative outcome of the simulations, we used $A = I$ and $R = \mathbf{0}$. Given the correspondence we've established, the KFU equations—which specify $\Psi_{t+1}$ and $\Sigma_{t+1}$ as a function of $c_t$, $e_t$, $\Psi_t$, and $\Sigma_t$—can be found in an introductory text (e.g., Maybeck, 1979).

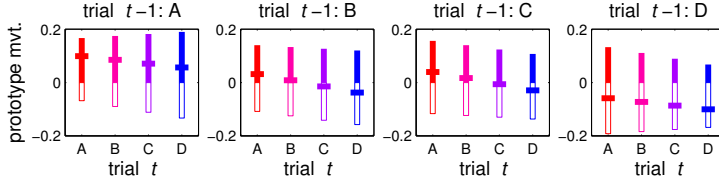

Figure 4: Change to a category prototype for each category following a trial of a given category. Solid (open) bars indicate trials in which the exemplar is larger (smaller) than the prototype.

## 2.2 Representation of the prototype

The prototype representation that we described is *localist*: there is a one-to-one correspondence between the prototype for each category $i$ and the random variable $D_i$. To select the appropriate prototype given a current category $c$, we defined the unary vector $\mathbf{h}_c$ and applied $\mathbf{h}_c$ as a linear transform on $\mathbf{D}$. The identical operations can be performed in conjunction with a *distributed* representation of the prototype. But we step back momentarily to motivate the distributed representation.

The localist representation suffers from a key weakness: it does not exploit interrelatedness constraints on category structure. The task given to experimental subjects specifies that there are four categories, and they have an ordering; the circle diameters associated with category A are smaller than the diameters associated with B, etc. Consequently, $d_A < d_B < d_C < d_D$. One might make a further assumption that the category prototypes are equally spaced. Exploiting these two sources of domain knowledge leads to the distributed representation of category structure.

A simple sort of distributed representation involves defining the prototype for category $i$ not as $d_i$ but as a linear function of an underlying two-dimensional state-space representation of structure. In this state space, $d_1$ indicates the distance between categories and $d_2$ an offset for all categories. This representation of state can be achieved by applying Equation 1 and defining $\mathbf{h}_c = (n_c, 1)$, where $n_c$ is the ordinal position of the category ($n_A = 1$, $n_B = 2$, etc.). We augment this representation with a bit of redundancy by incorporating not only the ordinal positions but also the reverse ordinal positions; this addition yields a symmetry in the representation between the two ends of the ordinal category scale. As a result of this augmentation, $\mathbf{d}$ becomes a three-dimensional state space, and $\mathbf{h}_c = (n_c, N + 1 - n_c, 1)$, where $N$ is the number of categories.

To summarize, both the localist and distributed representations posit the existence of a hidden-state space—unknown at the start of learning—that specifies category prototypes. The localist model assumes one dimension in the state space per prototype, whereas the distributed model assumes fewer dimensions in the state space—three, in our proposal—than there are prototypes, and computes the prototype location as a function of the state. Both localist and distributed representations assume a fixed, known $\{\mathbf{h}_c\}$ that specify the interpretation of the state space, or, in the case of the distributed model, the subject's domain knowledge about category structure.

## 3 Simulation Methodology

We defined a one-dimensional feature space in which categories A-D corresponded to the ranges [1, 2), [2, 3), [3, 4), and [4, 5), respectively. In the human experiment, responses were considered incorrect if they were smaller than A or larger than D; we call these two cases out-of-bounds-low (OOBL) and out-of-bounds-high (OOBH). OOBL and OOBH were treated as two additional categories, resulting in 6 categories altogether for the simulation. Subjects and the model were never asked to produce exemplars of OOBL or OOBH, but feedback was given if a response fell into these categories. As in the human experiment, our simulation involved 480 trials. We performed 100 replications of each simulation with identical initial conditions but different trial sequences, and averaged results over replications. All prototypes were initialized to have the same mean, 3.0, at the start of the simulation. Because subjects had some initial practice on the task before the start of the experimental trials, we provided the models with 12 initial trials of a categorization (not production) task, two for each of the 6 categories. (For the MLGA models, it was necessary to use a large step size on these trials to move the prototypes to roughly the correct neighborhood.)

To perform the production task, the models must generate an exemplar given a category. It seems natural to draw an exemplar from the distribution in Equation 2 for $P(E|C)$. However, this distribu-

tion reflects the full range of exemplars that lie within the category boundaries, and presumably in the production task, subjects attempt to produce a prototypical exemplar. Consequently, we exclude the intrinsic category variance, $v_c$, from Equation 2 in generating exemplars, leaving variance only via uncertainty about the prototype.

Each model involved selection of various parameters and initial conditions. We searched the parameter space by hand, attempting to find parameters that satisfied basic properties of the data: the accuracy and response variance in the first and second halves of the experiment. We report only parameters for the one model that was successful, the MLGA-Distrib: $\epsilon_\psi = 0.0075$, $\epsilon_\sigma = 1.5 \times 10^{-6}$ for off-diagonal terms and $1.5 \times 10^{-7}$ for diagonal terms (the gradient for the diagonal terms was relatively steep), $\Sigma_0 = 0.01I$, and for all categories $c$, $v_c = 0.4^2$.

## 4  Results

### 4.1  Push effect

The phenomenon that most clearly distinguishes the models is the push effect. The push effect is manifested in sequential-dependency functions, which plot the (relative) response on trial $t$ as a function of trial $t - 1$. As we explained using Figures 2a,b, the signature of the push effect is a negatively sloped line for each of the different trial $t$ target categories. The sequential-dependency functions for the four models are presented in Figures 2c-f. KFU-Local (Figure 2c) produces a flat line, indicating no push whatsoever. The explanation for this result is straightforward: the Kalman filter update alters only the variable that is responsible for the measurement (exemplar) obtained on that trial. That variable is the prototype of the target class $c$, $D_c$. We thought the lack of an interaction among the category prototypes might be overcome with KFU-Distrib, because with a distributed prototype representation, all of the state variables jointly determine the target category prototype. However, our intuition turned out to be incorrect. We experimented with many different representations and parameter settings, but KFU-Distrib consistently obtained flat or shallow positive sloping lines (Figure 2d).

MLGA-Local (Figure 2e) obtains a push effect for neighboring classes, but not distant classes. For example, examining the dashed magenta line, note that B is pushed away by A and C, but is not affected by D. MLGA-Local maximizes the likelihood of the target category both by pulling the class-conditional density of the target category toward the exemplar and by pushing the class-conditional densities of the other categories away from the exemplar. However, if a category has little probability mass at the location of the exemplar, the increase in likelihood that results from pushing it further away is negligible, and consequently, so is the push effect.

MLGA-Distrib obtains a lovely result (Figure 2f)—a negatively-sloped line, diagnostic of the push effect. The effect magnitude matches that in the human data (Figure 2a), and captures the key property that the push effect increases with the ordinal distance of the categories. We did not build a mechanism into MLGA-Distrib to produce the push effect; it is somewhat of an emergent property of the model. The state representation of MLGA-Distrib has three components: $d_1$, the weight of the ordinal position of a category prototype, $d_2$, the weight of the reverse ordinal position, and $d_3$, an offset. The last term, $d_3$, cannot be responsible for a push effect, because it shifts all prototypes equally, and therefore can only produce a flat sequential dependency function. Figure 4 helps provide an intuition how $d_1$ and $d_2$ work together to produce the push effect. Each graph shows the average movement of the category prototype (units on the y-axis are arbitrary) observed on trial $t$, for each of the four categories, following presentation of a given category on trial $t - 1$. Positve values on the y axis indicate increases in the prototype (movement to the right in Figure 1), and negative values decreases. Each solid vertical bar represents the movement of a given category prototype following a trial in which the exemplar is larger than its current prototype; each open vertical bar represents movement when the exemplar is to the left of its prototype. Notice that *all* category prototypes get larger or smaller on a given trial. But over the course of the experiment, the exemplar should be larger than the prototype as often as it is smaller, and the two shifts should sum together and partially cancel out. The result is the value indicated by the small horizontal bar along each line. The balance between the shifts in the two directions exactly corresponds to the push effect. Thus, the model produce a push-effect graph, but it is not truly producing a push effect as was originally conceived by the experimentalists. We are currently considering empirical consequences of this simulation result. Figure 5 shows a trial-by-trial trace from MLGA-Distrib.

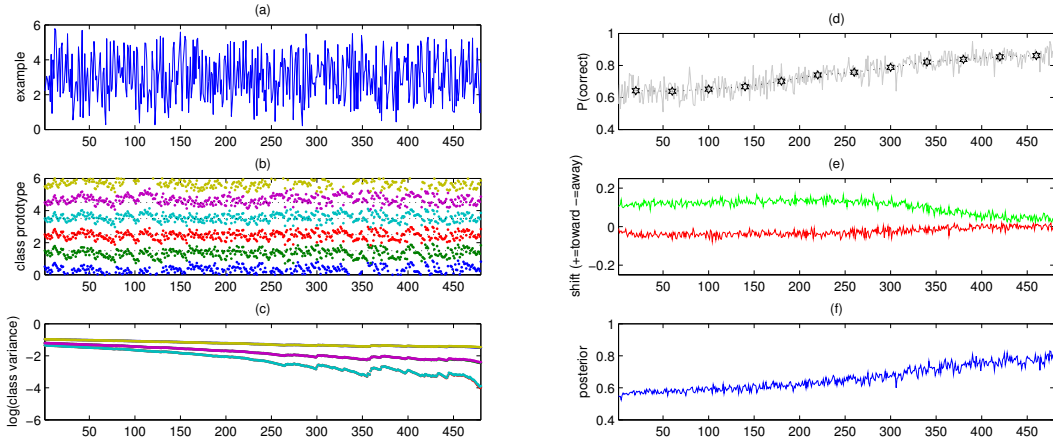

Figure 5: Trial-by-trial trace of MLGA-Distrib. (a) exemplars generated on one run of the simulation; (b) the mean and (c) variance of the class prototype distribution for the 6 classes on one run; (d) mean proportion correct over 100 replications of the simulation; (e) push and pull effects, as measured by changes to the prototype means: the upper (green) curve is the pull of the target prototype mean toward the exemplar, and the lower (red) curve is the push of the nontarget prototype means away from the exemplar, over 100 replications; (f) category posterior of the generated exemplar over 100 replications, reflecting gradient ascent in the posterior.

## 4.2 Other phenomena accounted for

MLGA-Distrib captures the other phenomena we listed at the outset of this paper. Like all of the other models, MLGA-Distrib readily produces a pull effect, which is shown in the movement of category prototypes in Figure 5e. More observably, a pull effect is manifested when two successive trials of the same category are positively correlated: when trial $t-1$ is to the left of the true category prototype, trial $t$ is likely to be to the left as well. In the human data, the correlation coefficient over the experiment is 0.524; in the model, the coefficient is 0.496. The explanation for the pull effect is apparent: moving the category prototype to the exemplar increases the category likelihood.

Although many learning effects in humans are based on error feedback, the experimental studies showed that push and pull effects occur even in the absence of errors, as they do in MLGA-Distrib. The model simply assumes that the target category it used to generate an exemplar is the correct category when no feedback to the contrary is provided. As long as the likelihood gradient is nonzero, category prototypes will be shifted.

Pull and push effects shrink over the course of the experiment in human studies, as they do in the simulation. Figure 5e shows a reduction in both pull and push, as measured by the shift of the prototype means toward or away from the exemplar. We measured the slope of MLGA-Distrib's push function (Figure 2f) for trials in the first and second half of the simulation. The slope dropped from $-0.042$ to $-0.025$, as one would expect from Figure 5e. (These slopes are obtained by combining responses from 100 replications of the simulation. Consequently, each point on the push function was an average over 6000 trials, and therefore the regression slopes are highly reliable.)

A quantitative, observable measure of pull is the standard deviation (SD) of responses. As push and pull effects diminish, SDs should decrease. In human subjects, the response SDs in the first and second half of the experiment are 0.43 and 0.33, respectively. In the simulation, the response SDs are 0.51 and 0.38. Shrink reflects the fact that the model is approaching a local optimum in log likelihood, causing gradients—and learning steps—to become smaller. Not all model parameter settings lead to shrink; as in any gradient-based algorithm, step sizes that are too large do not lead to converge. However, such parameter settings make little sense in the context of the learning objective.

## 4.3 Model predictions

MLGA-Distrib produces greater pull of the target category toward the exemplar than push of the neighboring categories away from the exemplar. In the simulation, the magnitude of the target pull— measured by the movement of the prototype mean—is 0.105, contrasted with the neighbor push,

which is 0.017. After observing this robust result in the simulation, we found pertinent experimental data. Using the categorization paradigm, Zotov et al. (2003) found that if the exemplar on trial $t$ is near a category border, subjects are more likely to produce an error if the category on trial $t-1$ is repeated (i.e., a pull effect just took place) than if the previous trial is of the neighboring category (i.e., a push effect), even when the distance between exemplars on $t-1$ and $t$ is matched. The greater probability of error translates to a greater magnitude of pull than push.

The experimental studies noted a phenomenon termed *snap back*. If the same target category is presented on successive trials, and an error is made on the first trial, subjects perform very accurately on the second trial, i.e., they generate an exemplar near the true category prototype. It appears as if subjects, realizing they have been slacking, reawaken and snap the category prototype back to where it belongs. We tested the model, but observed a sort of anti snap back. If the model made an error on the first trial, the mean deviation was larger—not smaller—on the second trial: 0.40 versus 0.32. Thus, MLGA-Distrib fails to explain this phenomenon. However, the phenomenon is not inconsistent with the model. One might suppose that on an error trial, subjects become more attentive, and increased attention might correspond to a larger learning rate on an error trial, which should yield a more accurate response on the following trial.

McLaren et al. (1995) studied a phenomenon in humans known as *peak shift*, in which subjects are trained to categorize unidimensional stimuli into one of two categories. Subjects are faster and more accurate when presented with exemplars far from the category boundary than those near the boundary. In fact, they respond more efficiently to far exemplars than they do to the category prototype. The results are characterized in terms of the prototype of one category being pushed away from the prototype of the other category. It seems straightforward to explain these data in MLGA-Distrib as a type of long-term push effect.

## 5    Related Work and Conclusions

Stewart, Brown, and Chater (2002) proposed an account of categorization context effects in which responses are based solely on the relative difference between the previous and present exemplars. No representation of the category prototype is maintained. However, classification based solely on relative difference cannot account for a diminished bias effects as a function of experience. A long-term stable prototype representation, of the sort incorporated into our models, seems necessary.

We considered four models in our investigation, and the fact that only one accounts for the experimental data suggests that the data are nontrivial. All four models have principled theoretical underpinnings, and they space they define may suggest other elegant frameworks for understanding mechanisms of category learning. The successful model, MLDA-Distrib, offers a deep insight into understanding multiple-category domains: category structure must be considered. MLGA-Distrib exploits knowledge available to subjects performing the task concerning the ordinal relationships among categories. A model without this knowledge, MLGA-Local, fails to explain data. Thus, the interrelatedness of categories appears to provide a source of constraint that individuals use in learning about the structure of the world.

**Acknowledgments**

This research was supported by NSF BCS 0339103 and NSF CSE-SMA 0509521. Support for the second author comes from an NSERC fellowship.

**References**

Jones, M. N., & Mewhort, D. J. K. (2003). Sequential contrast and assimilation effects in categorization of perceptual stimuli. Poster presented at the 44th Meeting of the Psychonomic Society. Vancouver, B.C.

Maybeck, P.S. (1979). *Stochastic models, estimation, and control, Volume I*. Academic Press.

McLaren, I. P. L., et al. (1995). Prototype effects and peak shift in categorization. *JEP:LMC*, *21*, 662–673.

Stewart, N. Brown, G. D. A., & Chater, N. (2002). Sequence effects in categorization of simple perceptual stimuli. *JEP:LMC*, *28*, 3–11.

Zotov, V., Jones, M. N., & Mewhort, D. J. K. (2003). Trial-to-trial representation shifts in categorization. Poster presented at the 13th Meeting of the Canadian Society for Brain, Behaviour, and Cognitive Science: Hamilton, Ontario.
